# Learning Depth from Single Monocular Images

**Ashutosh Saxena, Sung H. Chung, and Andrew Y. Ng**
Computer Science Department
Stanford University
Stanford, CA 94305
asaxena@stanford.edu,
{codedeft,ang}@cs.stanford.edu

## Abstract

We consider the task of depth estimation from a single monocular image. We take a supervised learning approach to this problem, in which we begin by collecting a training set of monocular images (of unstructured outdoor environments which include forests, trees, buildings, etc.) and their corresponding ground-truth depthmaps. Then, we apply supervised learning to predict the depthmap as a function of the image. Depth estimation is a challenging problem, since local features alone are insufficient to estimate depth at a point, and one needs to consider the global context of the image. Our model uses a discriminatively-trained Markov Random Field (MRF) that incorporates multiscale local- and global-image features, and models both depths at individual points as well as the relation between depths at different points. We show that, even on unstructured scenes, our algorithm is frequently able to recover fairly accurate depthmaps.

## 1 Introduction

Recovering 3-D depth from images is a basic problem in computer vision, and has important applications in robotics, scene understanding and 3-D reconstruction. Most work on visual 3-D reconstruction has focused on binocular vision (stereopsis) [1] and on other algorithms that require multiple images, such as structure from motion [2] and depth from defocus [3]. Depth estimation from a *single* monocular image is a difficult task, and requires that we take into account the global structure of the image, as well as use prior knowledge about the scene. In this paper, we apply supervised learning to the problem of estimating depth from single monocular images of unstructured outdoor environments, ones that contain forests, trees, buildings, people, buses, bushes, etc.

In related work, Michels, Saxena & Ng [4] used supervised learning to estimate 1-D distances to obstacles, for the application of autonomously driving a remote control car. Nagai et al. [5] performed surface reconstruction from single images for known, fixed, objects such as hands and faces. Gini & Marchi [6] used single-camera vision to drive an indoor robot, but relied heavily on known ground colors and textures. Shape from shading [7] offers another method for monocular depth reconstruction, but is difficult to apply to scenes that do not have fairly uniform color and texture. In work done independently of ours, Hoiem, Efros and Herbert (personal communication) also considered monocular 3-D reconstruction, but focused on generating 3-D graphical images rather than accurate metric depthmaps. In this paper, we address the task of learning full depthmaps from single images of unconstrained environments.

Markov Random Fields (MRFs) and their variants are a workhorse of machine learning, and have been successfully applied to numerous problems in which local features were insufficient and more contextual information had to be used. Examples include text segmentation [8], object classification [9], and image labeling [10]. To model spatial dependencies in images, Kumar and Hebert's Discriminative Random Fields algorithm [11] uses logistic regression to identify man-made structures in natural images. Because MRF learning is intractable in general, most of these model are trained using pseudo-likelihood.

Our approach is based on capturing depths and relationships between depths using an MRF. We began by using a 3-D distance scanner to collect training data, which comprised a large set of images and their corresponding ground-truth depthmaps. Using this training set, the MRF is discriminatively trained to predict depth; thus, rather than modeling the joint distribution of image features and depths, we model only the posterior distribution of the depths given the image features. Our basic model uses $L_2$ (Gaussian) terms in the MRF interaction potentials, and captures depths and interactions between depths at multiple spatial scales. We also present a second model that uses $L_1$ (Laplacian) interaction potentials. Learning in this model is approximate, but exact MAP posterior inference is tractable (similar to Gaussian MRFs) via linear programming, and it gives significantly better depthmaps than the simple Gaussian model.

## 2   Monocular Cues

Humans appear to be extremely good at judging depth from single monocular images. [12] This is done using monocular cues such as texture variations, texture gradients, occlusion, known object sizes, haze, defocus, etc. [4, 13, 14] For example, many objects' texture will look different at different distances from the viewer. Texture gradients, which capture the distribution of the direction of edges, also help to indicate depth.[1] Haze is another depth cue, and is caused by atmospheric light scattering.

Most of these monocular cues are "contextual information," in the sense that they are global properties of an image and cannot be inferred from small image patches. For example, occlusion cannot be determined if we look at just a small portion of an occluded object. Although local information such as the texture and color of a patch can give some information about its depth, this is usually insufficient to accurately determine its absolute depth. For another example, if we take a patch of a clear blue sky, it is difficult to tell if this patch is infinitely far away (sky), or if it is part of a blue object. Due to ambiguities like these, one needs to look at the *overall* organization of the image to determine depths.

## 3   Feature Vector

In our approach, we divide the image into small patches, and estimate a single depth value for each patch. We use two types of features: *absolute* depth features—used to estimate the absolute depth at a particular patch—and *relative* features, which we use to estimate relative depths (magnitude of the difference in depth between two patches). We chose features that capture three types of local cues: texture variations, texture gradients, and haze.

Texture information is mostly contained within the image intensity channel,[2] so we apply Laws' masks [15, 4] to this channel to compute the texture energy (Fig. 1). Haze is reflected in the low frequency information in the color channels, and we capture this by applying a local averaging filter (the first Laws mask) to the color channels. Lastly, to compute an

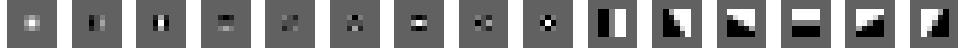

Figure 1: The convolutional filters used for texture energies and gradients. The first nine are 3x3 Laws' masks. The last six are the oriented edge detectors spaced at $30^0$ intervals. The nine Law's masks are used to perform local averaging, edge detection and spot detection.

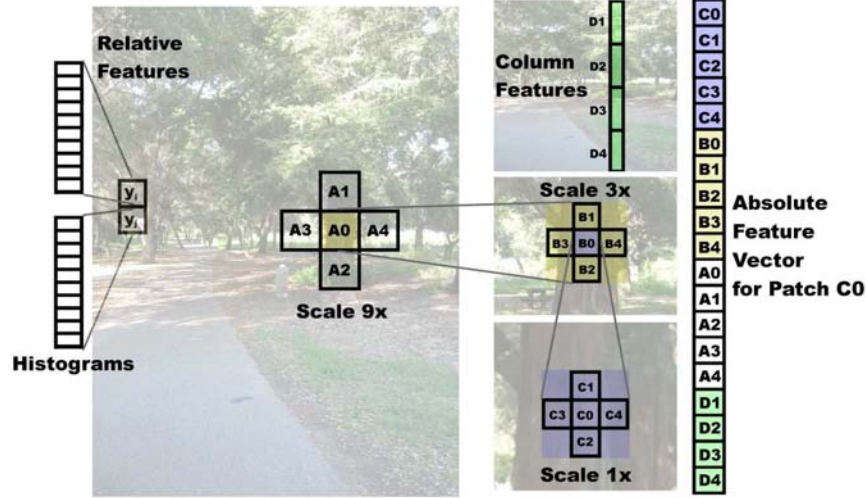

Figure 2: The absolute depth feature vector for a patch, which includes features from its immediate neighbors and its more distant neighbors (at larger scales). The relative depth features for each patch use histograms of the filter outputs.

estimate of texture gradient that is robust to noise, we convolve the intensity channel with six oriented edge filters (shown in Fig. 1).

### 3.1 Features for absolute depth

Given some patch $i$ in the image $I(x, y)$, we compute summary statistics for it as follows. We use the output of each of the 17 (9 Laws' masks, 2 color channels and 6 texture gradients) filters $F_n(x, y)$, $n = 1, ..., 17$ as: $E_i(n) = \sum_{(x,y)\in\text{patch}(i)} |I(x, y) * F_n(x, y)|^k$, where $k = \{1, 2\}$ give the sum absolute energy and sum squared energy respectively. This gives us an initial feature vector of dimension 34.

To estimate the absolute depth at a patch, local image features centered on the patch are insufficient, and one has to use more global properties of the image. We attempt to capture this information by using image features extracted at multiple scales (image resolutions). (See Fig. 2.) Objects at different depths exhibit very different behaviors at different resolutions, and using multiscale features allows us to capture these variations [16].[3] In addition to capturing more global information, computing features at multiple spatial scales also help accounts for different relative sizes of objects. A closer object appears larger in the image, and hence will be captured in the larger scale features. The same object when far away will be small and hence be captured in the small scale features. Such features may be strong indicators of depth.

To capture additional global features (e.g. occlusion relationships), the features used to predict the depth of a particular patch are computed from that patch as well as the four neighboring patches. This is repeated at each of the three scales, so that the feature vector

at a patch includes features of its immediate neighbors, and its far neighbors (at a larger scale), and its very far neighbors (at the largest scale), as shown in Fig. 2. Lastly, many structures (such as trees and buildings) found in outdoor scenes show vertical structure, in the sense that they are vertically connected to themselves (things cannot hang in empty air). Thus, we also add to the features of a patch additional summary features of the column it lies in.

For each patch, after including features from itself and its 4 neighbors at 3 scales, and summary features for its 4 column patches, our vector of features for estimating depth at a particular patch is $19 * 34 = 646$ dimensional.

## 3.2 Features for relative depth

We use a different feature vector to learn the dependencies between two neighboring patches. Specifically, we compute a histogram (with 10 bins) of each of the 17 filter outputs $|I(x, y) * F_n(x, y)|$, giving us a total of 170 features $y_i$ for each patch $i$. These features are used to estimate how the depths at two different locations are related. We believe that learning these estimates requires less global information than predicting absolute depth,[4] but more detail from the individual patches. Hence, we use as our relative depth features the differences between the histograms computed from two neighboring patches $y_{ij} = y_i - y_j$.

# 4    The Probabilistic Model

The depth of a particular patch depends on the features of the patch, but is also related to the depths of other parts of the image. For example, the depths of two adjacent patches lying in the same building will be highly correlated. We will use an MRF to model the relation between the depth of a patch and the depths of its neighboring patches. In addition to the interactions with the immediately neighboring patches, there are sometimes also strong interactions between the depths of patches which are not immediate neighbors. For example, consider the depths of patches that lie on a large building. All of these patches will be at similar depths, even if there are small discontinuities (such as a window on the wall of a building). However, when viewed at the smallest scale, some adjacent patches are difficult to recognize as parts of the same object. Thus, we will also model interactions between depths at multiple spatial scales.

Our first model will be a jointly Gaussian MRF. To capture the multiscale depth relations, let us define $d_i(s)$ as follows. For each of three scales $s = 1, 2, 3$, define $d_i(s + 1) = (1/5) \sum_{j \in N_s(i) \cup \{i\}} d_j(s)$. Here, $N_s(i)$ are the 4 neighbors of patch $i$ at scale $s$. I.e., the depth at a higher scale is constrained to be the average of the depths at lower scales. Our model over depths is as follows:

$$P(d|X; \theta, \sigma) = \frac{1}{Z} \exp \left( -\sum_{i=1}^{M} \frac{(d_i(1) - x_i^T \theta_r)^2}{2\sigma_{1r}^2} - \sum_{s=1}^{3} \sum_{i=1}^{M} \sum_{j \in N_s(i)} \frac{(d_i(s) - d_j(s))^2}{2\sigma_{2rs}^2} \right)$$
(1)

Here, $M$ is the total number of patches in the image (at the lowest scale); $x_i$ is the absolute depth feature vector for patch $i$; and $\theta$ and $\sigma$ are parameters of the model. In detail, we use different parameters $(\theta_r, \sigma_{1r}, \sigma_{2r})$ for each row in the image, because the images we consider are taken from a horizontally mounted camera, and thus different rows of the image have different statistical properties.[5] $Z$ is the normalization constant for the model.

We estimate the parameters $\theta_r$ in Eq. 1 by maximizing the conditional likelihood $p(d|X;\theta_r)$ of the training data. Since the model is a multivariate Gaussian, the maximum likelihood estimate of parameters $\theta_r$ is obtained by solving a linear least squares problem.

The first term in the exponent above models depth as a function of multiscale features of a single patch $i$. The second term in the exponent places a soft "constraint" on the depths to be smooth. If the variance term $\sigma_{2rs}^2$ is a fixed constant, the effect of this term is that it tends to smooth depth estimates across nearby patches. However, in practice the dependencies between patches are not the same everywhere, and our expected value for $(d_i - d_j)^2$ may depend on the features of the local patches.

Therefore, to improve accuracy we extend the model to capture the "variance" term $\sigma_{2rs}^2$ in the denominator of the second term as a linear function of the patches $i$ and $j$'s relative depth features $y_{ijs}$ (discussed in Section 3.2). We use $\sigma_{2rs}^2 = u_{rs}^T|y_{ijs}|$. This helps determine which neighboring patches are likely to have similar depths. E.g., the "smoothing" effect is much stronger if neighboring patches are similar. This idea is applied at multiple scales, so that we learn different $\sigma_{2rs}^2$ for the different scales $s$ (and rows $r$ of the image). The parameters $u_{rs}$ are chosen to fit $\sigma_{2rs}^2$ to the expected value of $(d_i(s) - d_j(s))^2$, with a constraint that $u_{rs} \geq 0$ (to keep the estimated $\sigma_{2rs}^2$ non-negative).

Similar to our discussion on $\sigma_{2rs}^2$, we also learn the variance parameter $\sigma_{1r}^2 = v_r^T x_i$ as a linear function of the features. The parameters $v_r$ are chosen to fit $\sigma_{1r}^2$ to the expected value of $(d_i(r) - \theta_r^T x_i)^2$, subject to $v_r \geq 0$.[6] This $\sigma_{1r}^2$ term gives a measure of the uncertainty in the first term, and depends on the features. This is motivated by the observation that in some cases, depth cannot be reliably estimated from the local features. In this case, one has to rely more on neighboring patches' depths to infer a patch's depth (as modeled by the second term in the exponent).

After learning the parameters, given a new test-set image we can find the MAP estimate of the depths by maximizing Eq. 1 in terms of $d$. Since Eq. 1 is Gaussian, $\log P(d|X;\theta,\sigma)$ is quadratic in $d$, and thus its maximum is easily found in closed form (taking at most 2-3 seconds per image, including feature computation time).

## 4.1 Laplacian model

We now present a second model that uses Laplacians instead of Gaussians to model the posterior distribution of the depths. Our motivation for doing so is three-fold. First, a histogram of the relative depths $(d_i - d_j)$ empirically appears Laplacian, which strongly suggests that it is better modeled as one. Second, the Laplacian distribution has heavier tails, and is therefore more robust to outliers in the image features and error in the training-set depthmaps (collected with a laser scanner; see Section 5.1). Third, the Gaussian model was generally unable to give depthmaps with sharp edges; in contrast, Laplacians tend to model sharp transitions/outliers better. Our model is as follows:

$$P(d|X;\theta,\lambda) = \frac{1}{Z}\exp\left(-\sum_{i=1}^{M}\frac{|d_i(1) - x_i^T\theta_r|}{\lambda_{1r}} - \sum_{s=1}^{3}\sum_{i=1}^{M}\sum_{j\in N_s(i)}\frac{|d_i(s) - d_j(s)|}{\lambda_{2rs}}\right) \quad (2)$$

Here, the parameters are the same as Eq. 1, except for the variance terms. Here, $\lambda_{1r}$ and $\lambda_{2rs}$ are the *Laplacian spread* parameters. Maximum-likelihood parameter estimation for the Laplacian model is not tractable (since the partition function depends on $\theta_r$). But by analogy to the Gaussian case, we approximate this by solving a linear system of equations $X_r\theta_r \approx d_r$ to minimize $L_1$ (instead of $L_2$) error. Here $X_r$ is the matrix of absolute-depth features. Following the Gaussian model, we also learn the Laplacian spread parameters in the denominator in the same way, except that the instead of estimating the expected value of $(d_i - d_j)^2$, we estimate the expected value of $|d_i - d_j|$. Even though maximum

likelihood parameter estimation for $\theta_r$ is intractable in the Laplacian model, given a new test-set image, MAP inference for the depths $d$ is tractable. Specifically, $P(d|X; \theta, \lambda)$ is easily maximized in terms of $d$ using linear programming.

**Remark.** We can also extend these models to combine Gaussian and Laplacian terms in the exponent, for example by using a $L_2$ norm term for absolute depth, and a $L_1$ norm term for the interaction terms. MAP inference remains tractable in this setting, and can be solved using convex optimization as a QP (quadratic program).

## 5  Experiments

### 5.1  Data collection

We used a 3-D laser scanner to collect images and their corresponding depthmaps. The scanner uses a SICK 1-D laser range finder mounted on a motor to get 2D scans. We collected a total of 425 image+depthmap pairs, with an image resolution of 1704x2272 and a depthmap resolution of 86x107. In the experimental results reported here, 75% of the images/depthmaps were used for training, and the remaining 25% for hold-out testing. Due to noise in the motor system, the depthmaps were not perfectly aligned with the images, and had an alignment error of about 2 depth patches. Also, the depthmaps had a maximum range of 81m (the maximum range of the laser scanner), and had minor additional errors due to reflections and missing laser scans. Prior to running our learning algorithms, we transformed all the depths to a log scale so as to emphasize multiplicative rather than additive errors in training. In our earlier experiments (not reported here), learning using linear depth values directly gave poor results.

### 5.2  Results

We tested our model on real-world test-set images of forests (containing trees, bushes, etc.), campus areas (buildings, people, and trees), and indoor places (such as corridors). The algorithm was trained on a training set comprising images from *all* of these environments. Table 1 shows the test-set results when using different feature combinations. We see that using multiscale and column features significantly improves the algorithm's performance.

Including the interaction terms further improved its performance, and the Laplacian model performs better than the Gaussian one. Empirically, we also observed that the Laplacian model does indeed give depthmaps with significantly sharper boundaries (as in our discussion in Section 4.1; also see Fig. 3). Table 1 shows the errors obtained by our algorithm on a variety of forest, campus, and indoor images. The results on the test set show that the algorithm estimates the depthmaps with a average error of $0.132$ orders of magnitude. It works well even in the varied set of environments as shown in Fig. 3 (last column). It also appears to be very robust towards variations caused by shadows.

Informally, our algorithm appears to predict the relative depths of objects quite well (i.e., their relative distances to the camera), but seems to make more errors in absolute depths. Some of the errors can be attributed to errors or limitations of the training set. For example, the training set images and depthmaps are slightly misaligned, and therefore the edges in the learned depthmap are not very sharp. Further, the maximum value of the depths in the training set is 81m; therefore, far-away objects are all mapped to the one distance of 81m.

Our algorithm appears to incur the largest errors on images which contain very irregular trees, in which most of the 3-D structure in the image is dominated by the shapes of the leaves and branches. However, arguably even human-level performance would be poor on these images.

## 6  Conclusions

We have presented a discriminatively trained MRF model for depth estimation from single monocular images. Our model uses monocular cues at multiple spatial scales, and also

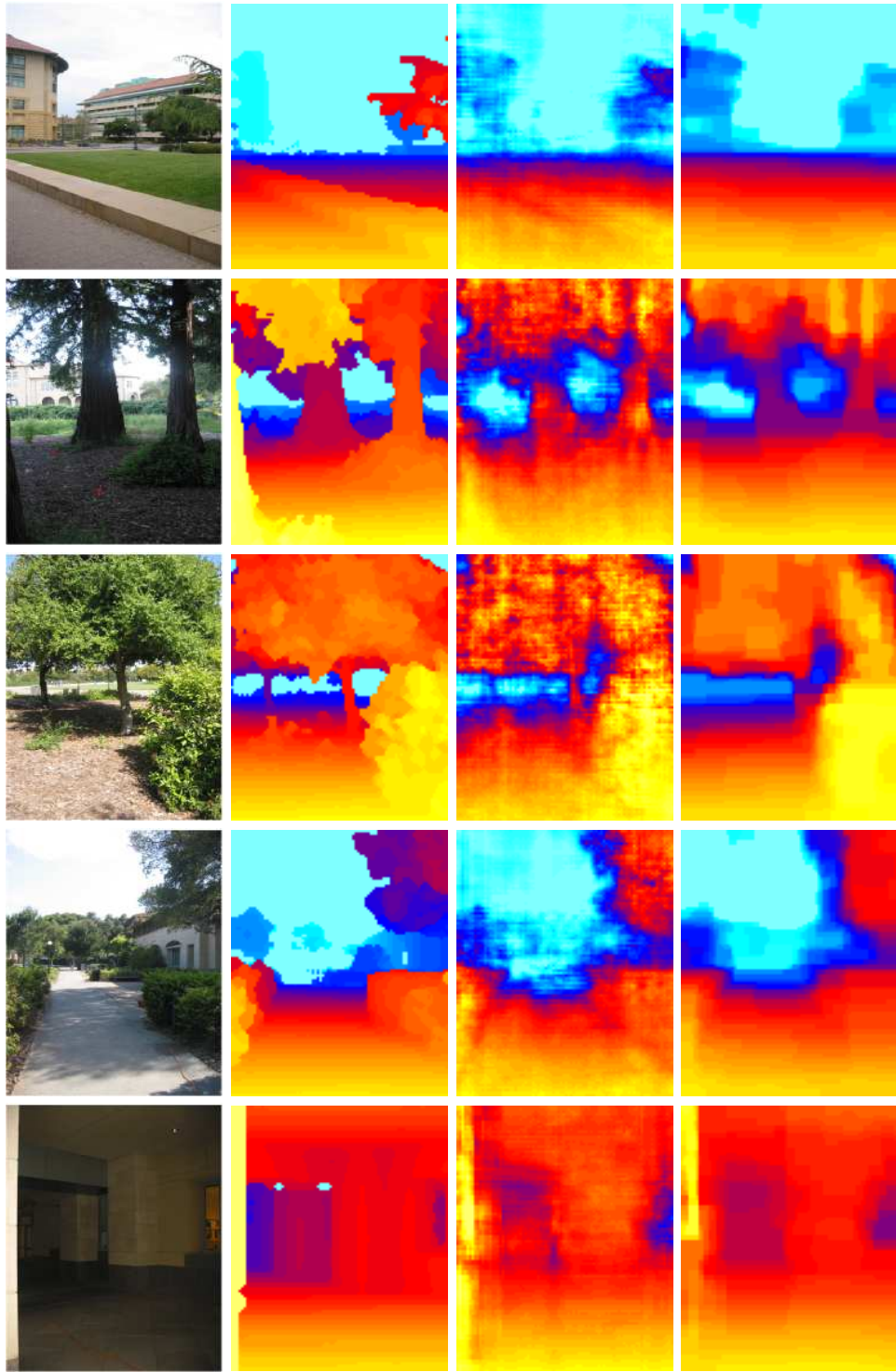

Figure 3: Results for a varied set of environments, showing original image (column 1), ground truth depthmap (column 2), predicted depthmap by Gaussian model (column 3), predicted depthmap by Laplacian model (column 4). (**Best viewed in color**)

Table 1: Effect of multiscale and column features on accuracy. The average absolute errors (RMS errors gave similar results) are on a log scale (base 10). $H_1$ and $H_2$ represent summary statistics for $k = 1, 2$. $S_1$, $S_2$ and $S_3$ represent the 3 scales. $C$ represents the column features. Baseline is trained with only the bias term (no features).

| FEATURE | ALL | FOREST | CAMPUS | INDOOR |
|---|---|---|---|---|
| BASELINE | .295 | .283 | .343 | .228 |
| GAUSSIAN ($S_1,S_2,S_3$, $H_1,H_2$,*no neighbors*) | .162 | .159 | .166 | .165 |
| GAUSSIAN ($S_1$, $H_1,H_2$) | .171 | .164 | .189 | .173 |
| GAUSSIAN ($S_1,S_2$, $H_1,H_2$) | .155 | .151 | .164 | .157 |
| GAUSSIAN ($S_1$, $S_2,S_3$, $H_1,H_2$) | .144 | .144 | .143 | .144 |
| GAUSSIAN ($S_1,S_2,S_3$, $C$, $H_1$) | .139 | .140 | .141 | .122 |
| GAUSSIAN ($S_1,S_2,S_3$, $C$, $H_1,H_2$) | .133 | .135 | .132 | .124 |
| LAPLACIAN | .132 | .133 | .142 | .084 |

incorporates interaction terms that model relative depths, again at different scales. In addition to a Gaussian MRF model, we also presented a Laplacian MRF model in which MAP inference can be done efficiently using linear programming. We demonstrated that our algorithm gives good 3-D depth estimation performance on a variety of images.

**Acknowledgments**
We give warm thanks to Jamie Schulte, who designed the 3-D scanner, for help in collecting the data used in this work. We also thank Larry Jackel for helpful discussions. This work was supported by the DARPA LAGR program under contract number FA8650-04-C-7134.

## Footnotes

[1]For example, a tiled floor with parallel lines will appear to have tilted lines in an image. The distant patches will have larger variations in the line orientations, and nearby patches will have smaller variations in line orientations. Similarly, a grass field when viewed at different distances will have different texture gradient distributions.

[2]We represent each image in YCbCr color space, where Y is the intensity channel, and Cb and Cr are the color channels.

[3]For example, blue sky may appear similar at different scales; but textured grass would not.

[4]For example, given two adjacent patches of a distinctive, unique, color and texture, we may be able to safely conclude that they are part of the same object, and thus that their depths are close, even without more global features.

[5]For example, a blue patch might represent sky if it is in upper part of image, and might be more likely to be water if in the lower part of the image.

[6]The absolute depth features $x_{ir}$ are non-negative; thus, the estimated $\sigma_{1r}^2$ is also non-negative.

# References

[1] D. Scharstein and R. Szeliski. A taxonomy and evaluation of dense two-frame stereo correspondence algorithms. *Int'l Journal of Computer Vision*, 47:7–42, 2002.

[2] David A. Forsyth and Jean Ponce. *Computer Vision : A Modern Approach*. Prentice Hall, 2003.

[3] S. Das and N. Ahuja. Performance analysis of stereo, vergence, and focus as depth cues for active vision. *IEEE Trans Pattern Analysis & Machine Intelligence*, 17:1213–1219, 1995.

[4] J. Michels, A. Saxena, and A.Y. Ng. High speed obstacle avoidance using monocular vision and reinforcement learning. In *ICML*, 2005.

[5] T. Nagai, T. Naruse, M. Ikehara, and A. Kurematsu. Hmm-based surface reconstruction from single images. In *Proc IEEE Int'l Conf Image Processing*, volume 2, 2002.

[6] G. Gini and A. Marchi. Indoor robot navigation with single camera vision. In *PRIS*, 2002.

[7] M. Shao, T. Simchony, and R. Chellappa. New algorithms from reconstruction of a 3-d depth map from one or more images. In *Proc IEEE CVPR*, 1988.

[8] J. Lafferty, A. McCallum, and F. Pereira. Discriminative fields for modeling spatial dependencies in natural images. In *ICML*, 2001.

[9] K. Murphy, A. Torralba, and W.T. Freeman. Using the forest to see the trees: A graphical model relating features, objects, and scenes. In *NIPS 16*, 2003.

[10] Xuming He, Richard S. Zemel, and Miguel A. Carreira-Perpinan. Multiscale conditional random fields for image labeling. In *proc. CVPR*, 2004.

[11] S. Kumar and M. Hebert. Discriminative fields for modeling spatial dependencies in natural images. In *NIPS 16*, 2003.

[12] J.M. Loomis. Looking down is looking up. *Nature News and Views*, 414:155–156, 2001.

[13] B. Wu, T.L. Ooi, and Z.J. He. Perceiving distance accurately by a directional process of integrating ground information. *Letters to Nature*, 428:73–77, 2004.

[14] P. Sinha I. Blthoff, H. Blthoff. Top-down influences on stereoscopic depth-perception. *Nature Neuroscience*, 1:254–257, 1998.

[15] E.R. Davies. Laws' texture energy in TEXTURE. In *Machine Vision: Theory, Algorithms, Practicalities 2nd Edition*. Academic Press, San Diego, 1997.

[16] A.S. Willsky. Multiresolution markov models for signal and image processing. *IEEE*, 2002.
